# Detecting Humans via Their Pose

**Alessandro Bissacco**
Computer Science Department
University of California, Los Angeles
Los Angeles, CA 90095
bissacco@cs.ucla.edu

**Ming-Hsuan Yang**
Honda Research Institute
800 California Street
Mountain View, CA 94041
mhyang@ieee.org

**Stefano Soatto**
Computer Science Department
University of California, Los Angeles
Los Angeles, CA 90095
soatto@cs.ucla.edu

## Abstract

We consider the problem of detecting humans and classifying their pose from a single image. Specifically, our goal is to devise a statistical model that simultaneously answers two questions: 1) is there a human in the image? and, if so, 2) what is a low-dimensional representation of her pose? We investigate models that can be learned in an unsupervised manner on unlabeled images of human poses, and provide information that can be used to match the pose of a new image to the ones present in the training set. Starting from a set of descriptors recently proposed for human detection, we apply the Latent Dirichlet Allocation framework to model the statistics of these features, and use the resulting model to answer the above questions. We show how our model can efficiently describe the space of images of humans with their pose, by providing an effective representation of poses for tasks such as classification and matching, while performing remarkably well in human/non human decision problems, thus enabling its use for human detection. We validate the model with extensive quantitative experiments and comparisons with other approaches on human detection and pose matching.

## 1   Introduction

Human detection and localization from a single image is an active area of research that has witnessed a surge of interest in recent years [9, 18, 6]. Simply put, given an image, we want to devise an automatic procedure that locates the regions that contain human bodies in arbitrary pose. This is hard because of the wide variability that images of humans exhibit. Given that it is impractical to explicitly model nuisance factors such as clothing, lighting conditions, viewpoint, body pose, partial and/or self-occlusions, one can learn a descriptive model of human/non human statistics. The problem then reduces to a binary classification task for which we can directly apply general statistical learning techniques. Consequently, the main focus of research on human detection so far has been on deriving a suitable representation [9, 18, 6], i.e. one that is most insensitive to typical appearance variations, so that it provides good features to a standard classifier.

Recently local descriptors based on histograms of gradient orientations such as [6] have proven to be particularly successful for human detection tasks. The main idea is to use distributions of gradient orientations in order to be insensitve to color, brightness and contrast changes and, to some extent, local deformations. However, to account for more macroscopic variations, due for example to changes in pose, a more complex statistical model is warranted. We show how a special class of hierarchical Bayesian processes can be used as generative models for these features and applied to the problem of detection and pose classification.

This work can be interpreted as an attempt to bridge the gap between the two related problems of human detection and pose estimation in the literature. In human detection, since a simple yes/no answer is required, there is no need to introduce a complex model with latent variables associated to physical quantities. In pose estimation, on the other hand, the goal is to infer these quantities and therefore a full generative model is a natural approach. Between these extremes lies our approach. We estimate a probabilistic model with a set of latent variables, which do not necessarily admit a direct interpretation in terms of configurations of objects in the image. However, these quantities are instrumental to both human detection and the pose classification problem.

The main difficulty is in the representation of the pose information. Humans are highly articulated objects with many degrees of freedom, which makes defining pose classes a remarkably difficult problem. Even with manual labeling, how does one judge the distance between two poses or cluster them? In such situations, we believe that the only avenue is an unsupervised method. We propose an approach which allows for unsupervised clustering of images of humans and provides a low dimensional representation encoding essential information on their pose. The chief difference with standard clustering or dimensionality reduction techniques is that we derive a full probabilistic framework, which provides principled ways to combine and compare different models, as required for tasks such as human detection, pose classification and matching.

## 2    Context and Motivation

The literature on human detection and pose estimation is too broad for us to review here. So we focus on the case of a single image, neglecting scenarios where temporal information or a background model are available and  effective algorithms based on silhouettes [20, 12, 1] or motion patterns [18] can be applied.

Detecting humans and estimating poses from single images is a fundamental problem with a range of sensible applications, such as image retrieval and understanding. It makes sense to tackle this problem as we know humans are capable of telling the locations and poses of people from the visual information contained in photographs. The question is how to represent such information, and the answer we give constitutes the main novelty of this work.

Numerous representation schemes have been exploited for human detection, e.g., Haar wavelets [18], edges [9], gradient orientations [6], gradients and second derivatives [19] and regions from image segmentation [15]. With these representations, algorithms have been applied for the detection process such as template matching [9], support vector machine [19, 6], Adaboost [18], and grouping [15], to name a few. Most approaches to pose estimation are based on body part detectors, using either edge, shape, color and texture cues [7, 21, 15], or learned from training data [19]. The optimal configuration of the part assembly is then computed using dynamic programming as first introduced in [7], or by performing inference on a generative probabilistic model, using either Data Driven Markov Chain Monte Carlo, Belief Propagation or its non-Gaussian extensions [21].

These works focus on only one of the two problems, either detection or pose estimation. Our approach is different, in that our goal is to extract more information than a simple yes/no answer, while at the same time not reaching the full level of detail of determining the precise location of all body parts. Thus we want to simultaneously perform detection and pose classification, and we want to do it in an unsupervised manner. In this aspect, our work is related to the constellation models of Weber et al. [23], although we do not have an explicit decomposition of the object in parts.

We start from the representation [6] based on gradient histograms recently applied to human detection with excellent results, and derive a probabilistic model for it. We show that with this model one can successfully detect humans and classify their poses. The statistical tools used in this work, Latent Dirichlet Allocation (LDA) [3] and related algorithms [5, 4], have been introduced in the text analysis context and recently applied to the problem of recognition of object and action classes [8, 22, 2, 16]. Contrary to most approaches (all but [8]) where the image is treated as a "bag of features" and all spatial information is lost, we encode the location and orientation of edges in the basic elements (words) so that this essential information is explicitly represented by the model.

## 3    A Probabilistic Model for Gradient Orientations

We first describe the features that we use as the basic representations of images, and then propose a probabilistic model with its application to the feature generation process.

## 3.1 Histogram of Oriented Gradients

Local descriptors based on gradient orientations are one of the most successful representations for image-based detection and matching, as was firstly demonstrated by Lowe in [14]. Among the various approaches within this class, the best performer for humans appears to be [6]. This descriptor is obtained by computing weighted histograms of gradient orientations over a grid of spatial neighborhoods (cells), which are then grouped in overlapping regions (blocks) and normalized for brightness and contrast changes.

Assume that we are given a patch of $64 \times 128$ pixels, we divide the patch into cells of $8 \times 8$ pixels, and for each cell a gradient orientation histogram is computed. The histogram represents a quantization in 9 bins of gradient orientations in the range $0° - 180°$. Each pixel contributes to the neighboring bins, both in orientation and space, by an amount proportional to the gradient magnitude and linearly decreasing with the distance from the bin center. These cells are grouped in $2 \times 2$ blocks, and the contribution of each pixel is also weighted by a Gaussian kernel with $\sigma = 8$, centered in the block. Finally the vectors $\mathbf{v}$ of cell histograms within one block are normalized in $L_2$ norm: $\bar{\mathbf{v}} = \mathbf{v}/(||\mathbf{v}||_2 + \epsilon)$. The final descriptor is a collection of histograms from overlapping blocks (each cell shared by 4 blocks).

The main characteristic of such a representation is robustness to local deformations, illumination changes and, to a limited extent, viewpoint and pose changes due to coarsening of the histograms. In order to handle the larger variations typical of human body images, we need to complement this representation with a model. We propose a probabilistic model that can accurately describe the generation process of these features.

## 3.2 Latent Dirichlet Allocation

Latent Dirichlet Allocation (LDA) [3] is a hierachical model for sparse discrete mixture distributions, where the basic elements (words) are sampled from a mixture of component distributions, and each component defines a discrete distribuition over the set of words.

We are given a collection of documents where words $w$, the basic units of our data, take values in a dictionary of $W$ unique elements $w \in \{ 1, \cdots, W \}$. A document $\mathbf{w} = ( w_1, w_2, \cdots, w_W )$ is a collection of word counts $w_j$: $\sum_{j=1}^{W} w_j = N$. The standard LDA model does not include the distribution of $N$, so it can be omitted in what follows. The corpus $\mathcal{D} = \{ \mathbf{w_1}, \mathbf{w_2}, \cdots, \mathbf{w_M} \}$ is a collection of $M$ documents.

The LDA model introduces a set of $K$ latent variables, called topics. Each word in the document is assumed to be generated by one of the topics. Under this model, the generative process for each document $\mathbf{w}$ in the corpus is as follows:

1. Choose $\theta \sim \text{Dirichlet}(\alpha)$.
2. For each word $j = 1, \cdots, W$ in the dictionary, choose a word count $w_j \sim p(w_j|\theta, \beta)$.

where the word counts $w_j$ are drawn from a discrete distribution conditioned on the topic proportions $\theta$: $p(w_j|\theta, \beta) = \beta_j.\theta$. Recently several variants to this model have been developed, notably the Multinomial PCA [4], where the discrete distributions are replaced by multinomials, and the Gamma-Poisson process [5], where the number of words $\theta_i$ from each component are independent Gamma samples and $p(w_j|\theta, \beta)$ is Poisson. The hyperparameter $\alpha \in \mathcal{R}_+^K$ represents the prior on the topic distribution, $\theta \in \mathcal{R}_+^K$ are the topic proportions, and $\beta \in \mathcal{R}_+^{W \times K}$ are the parameters of the word distributions conditioned on topics. In the context of this work, words correspond to oriented gradients, and documents as well as corpus correspond to images and a set of images respectively. The topic derived by the LDA model is the pose of interest in this work. Here we can safely assume that the topic distributions $\beta$ are deterministic parameters, later for the purpose of inference we will treat them as random variables and assign them a Dirichlet prior: $\beta_{.k} \sim \text{Dirichlet}(\eta)$, where $\beta_{.k}$ denotes the $k$-th column of $\beta$.

Then the likelihood of a document $\mathbf{w}$ is:

$$p(\mathbf{w}|\alpha, \beta) = \int p(\theta|\alpha) \prod_{n=1}^{W} p(w_n|\theta, \beta)d\theta \tag{1}$$

where documents are represented as a continuous mixture distribution. The advantage over standard mixture of discrete distributions [17], is that we allow each document to be generated by more than one topic.

### 3.3 A Bayesian Model for Gradient Orientation Histograms

Now we can show how the described two-level Bayesian process finds a natural application in modeling the spatial distribution of gradient orientations. Here we consider the histogram of oriented gradients [6] as the basic feature from which we build our generative model, but let us point out that the framework we introduce is more general and can be applied to any descriptor based on histograms[1]. In this histogram descriptor, we have that each bin represents the intensity of the gradient at a particular location, defined by a range of orientations and a local neighborhood (cell). Thus the bin height denotes the strength and number of the edges in the cell.

The first thing to notice in deriving a generative models for this class of features is that, since they represent a weighted histogram, they have non-negative elements. Thus a proper generative model for these descriptors imposes non-negativity constraints. As we will see in the experiments, a linear approach such as Non-negative Matrix Factorization [13] leads to extremely poor performance, probably due to the high curvature of the space. On the opposite end, representing the nonlinearity of the space with a set of samples by Vector Quantization is feasible only using a large number of samples, which is against our goal of deriving an economical representation of the pose.

We propose using the Latent Dirichlet Allocation model to represent the statistics of the gradient orientation features. In order to do so we need to quantize feature values. While not investigated in the original paper [6], quantization is common practice for similar histogram-based descriptors, such as [14]. We tested the effect of quantization on the performance of the human detector based on Histogram of Oriented Gradient descriptors and linear Support Vector Machines described in [6]. As evident in Figure 1, with 16 or more discrete levels we practically obtain the same performance as with the original continuous descriptors. Thus in what follows we can safely assume that the basic features are collections of small integers, the histogram bin counts $w_j$.

Thus, if we quantize histogram bins and assign a unique word to each bin, we obtain a representation for which we can directly apply the LDA framework. Analogous to document analysis, an orientation histogram computed on an image patch is a document $\mathbf{w}$ represented as a bag of words $(\ w_1, \cdots, w_W\ )$, where the word counts $w_j$ are the bin heights. We assume that such a histogram is generated by a mixture of basic components (topics), where each topic $z$ induces a discrete distribution $p(r|\beta_{.z})$ on bins representing a typical configuration of edges common to a class of elements in the dataset. By summing the contributions from each topic we obtain the total count $w_j$ for each bin, distributed according to $p(w_j|\theta, \beta)$.

The main property of such feature formation process, desirable for our applications, is the fact that topics combine additively. That is, the same bin may have contributions from multiple topics, and this models the fact that the bin height is the count of edges in a neighborhood which may include parts generated by different components. Finally, let us point out that by assigning a unique word to each bin we model spatial information, encoded in the word identity, whereas most previous approaches (e.g. [22]) using similar probabilistic models for object class recognition did not exploit this kind of information.

## 4 Probabilistic Detection and Pose Estimation

The first application of our approach is human detection. Notice that our main goal is to develop a model to represent the statistics of images for human pose classification. We use the human detection problem as a convenient testbed for validating the goodness of our representation, since for this application large labelled datasets and efficient algorithms are available. By no means we intend to compete with state-of-the-art discriminative approaches for human detection alone, which are optimized to represent the decision boundary and thus are supposed to perform better than generative approaches in binary classification tasks. However, if the generative model is good at capturing the statistics of human images we expect it to perform well also in discriminating humans from the background.

In human detection, given a set of positive and negative examples and a previously unseen image $I_{new}$, we are asked to choose between two hypotheses: either it contains a human or it is a background image. The first step is to compute the gradient histogram representation $\mathbf{w}(I)$ for the test and training images. Then we learn a model for humans and background images and use a threshold

on the likelihood ratio[2] for detection:

$$L = \frac{P(\mathbf{w}(I_{new})|\text{Human})}{P(\mathbf{w}(I_{new})|\text{Background})} \qquad (2)$$

For the the LDA (and related models [5, 4], the likelihoods $p(\mathbf{w}(\mathbf{I})|\alpha, \beta)$ are computed as in (1), where $\alpha, \beta$ are model parameters and can be learned from data. In practice, we can assume $\alpha$ is known and compute an estimate of $\beta$ from the training corpus. In doing so, we can choose from two main inference algorithms: mean field or variational inference [3] and Gibbs sampling [10]. Mean field algorithms provide a lower bound on the likelihood, while Gibbs sampling gives statistics based on a sequential sampling scheme. As shown in Figure 1, in our experiments Gibbs sampling exhibited superior performance over mean field in terms of classification accuracy. We have experimented with two variations, a direct method and Rao-Blackwellised sampling (see [4] for details). Both methods gave similar performance, here we report the results obtained using the direct method, whose main iteration is as follows:

1. For each document $\mathbf{w}_i = (w_{i,1}, \cdots, w_{i,W})$:
   First sample $\theta^{(i)} \sim p(\theta|\mathbf{w}_i, \alpha, \beta)$, and then sample $v_{j.}^{(i)} \sim \text{Multinomial}(\beta_{j.}\theta^{(i)}, w_{i,j})$
2. For each topic $k$:
   Sample $\beta_{.k} \sim \text{Dirichlet}(\sum_i v_{.k}^{(i)} + \eta)$

In pose classification, we start from a set of unlabeled training examples of human poses and learn the topic distribution $\beta$. This defines a probabilistic mapping to the topic variables, which can be seen as an economical representation encoding essential information of the pose. That is, from a image $I_{new}$, we estimate the topic proportions $\hat{\theta}(I_{new})$ as:

$$\hat{\theta}(I_{new}) = \int \theta p(\theta|w(I_{new}), \alpha, \beta) d\theta \qquad (3)$$

Pose information can be recovered by matching the new image $I_{new}$ to an image $I$ in the training set. For matching, ideally we would like to compute the matching score as $S_{opt}(I, I_{new}) = P(\mathbf{w}(I_{new})|\mathbf{w}(I), \alpha, \beta)$, i.e. the posterior probability of the test image $I_{new}$ given the training image $I$ and the model $\alpha, \beta$. However this would be computationally expensive as for each pair $I, I_{new}$ it requires computing an expectation of the form (3), thus we opted for a suboptimal solution. For each training document $I$, in the learning step we compute the posterior topic proportions $\hat{\theta}(I)$ as in (3). Then the matching score $S$ between $I_{new}$ and $I$ is given by the dot product between the two vectors $\hat{\theta}(I)$ and $\hat{\theta}(I_{new})$:

$$S(I, I_{new}) = <\hat{\theta}(I), \hat{\theta}(I_{new})> \qquad (4)$$

The computation of this score requires only a dot product between low dimensional unit vectors $\hat{\theta}$, so our approach represent an efficient method for matching and clustering poses in large datasets.

## 5   Experiments

We first tested the efficacy of our model for the human detection task. We used the dataset provided by [6], consisting of 2340 $64 \times 128$ images of pedestrians in various configurations and 1671 images of outdoor scenes not containing humans. We collected negative examples by random sampling 10 patches from each of the first 1218 non-human images. These, together with 1208 positive examples and their left-right reflections, constituted our first training set. We used the learned model to classify remaining 1132 positive and on 5889 patches randomly extracted from the residual background images.

We first computed the histograms of oriented gradients from the image patches following the procedure outlined in Section 3.1. These feature are quantized so that they can be represented by our discrete stochastic model.

We tested the effect of different quantization levels on the performances of the boosted SVM classifier [6]: a initial training on the provided dataset is followed by a boosting round where the trained classifier is applied to the background images to find false positive; these hard examples are then added to for a second training of the classifier. As Figure 1 shows, the effect of quantization is significant only if we use less than 4 bits. Therefore, we chose to discretize the features to 16 quantization levels.

Given the number of topics $K$ and the prior hyperparameters $\alpha, \eta$, we learned topic distributions $\beta$ and topic proportions $\hat{\theta}(I)$ using either Gibbs sampling or Mean Field. We tested both Gamma [5] and Dirichlet [3, 4] distributions for topic priors, obtaining best results with the multinomial model [4] with scalar priors $\alpha_i = a, \eta_i = b$, in these experiments $a = 2/K$ and $b = 0.5$.

The number of topics $K$ is an important parameter that should be carefully chosen based on considerations on modeling power and complexity. With a higher number of topics we can more accurately fit the data, which can be measured by the increase in the likelihood of the training set. This does not come for free: we have a larger number of parameters and an increased computational cost for learning. Eventually, an excessive topic number causes overfitting, which can be measured as the likelihood in the test dataset decreases. For the INRIA data, experimental evaluations suggested that a good tradeoff is obtained with $K = 24$.

We learned two models, one for positive and one for negative examples. For learning we run the Gibbs sampling algorithm described in Section 4 for a total number of 300 samples per document, including 50 samples to compute the likelihoods (1). We also trained the model using the Mean Field approximation, but as we can see in Figures 1 and 4 the results using Gibbs sampling are better. For details on the implementation we refer to [4]. We then obtain a detector by computing the likelihood ratio (2) and comparing it with a threshold.

In Figure 1 we show the performances of our detector on the INRIA dataset, where for the sake of comparison with other approaches boosting is not performed. We show the results for:

- Linear SVM classifier: Trained as described, using the SVMLight software package.
- Vector Quantization: Positive and negative models learned as collections of $K$ clusters using the K-Means algorithm. Then the decision rule is Nearest Neighbor, that is whether the closest cluster belongs to positive or negative model.
- Non-negative Matrix Factorization: Feature vectors are collected in a matrix $V$, and the factorization that minimizes $||Y - WH||_2^2$ with $W, H$ nonnegative is computed using the multiplicative update algorithm of [13]. Using an analogy with the LDA model, the columns of $W$ contain the topic distributions, while the columns of $H$ represent the component weights. A classifier is obtained as the difference of the residuals of the feature projections on the positive and negative models.

From the plot we see how the results of our approach are comparable with the performance of the Linear SVM, while being far superior to the other generative approaches. We would like to stress that a sole comparison on detection performance with state-of-the discriminative classifiers would be inappropriate, since our model targets pose classification which is harder than binary detection. A fair comparison should divide the dataset in classes and compare our model with a multiclass classifier. But then we would face the difficult problem of how to label human poses.

For the experiments on pose classification and matching, we used the CMU Mobo dataset [11]. It consists of sequences of subjects performing different motion patterns, each sequence taken from 6 different views. In the experiments we used 22 sequences of fast walking motion, picking the first 100 frames from each sequence.

In the first experiment we trained the model with all the views and set the number of topics equal to the number of views, $K = 6$. As expected, each topic distribution represents a view and by assigning every image $I$ to the topic $k$ with highest proportion $k = \arg\max_k \hat{\theta}_k(I)$ we correctly associated all the images from the same view to the same topic.

To obtain a more challenging setup, we restricted to a single view and tested the classification performance of our approach in matching poses. We learned a model with $K = 8$ topics from 16 training sequences, and used the remaining 6 for testing. In Figure 2 we show sample topics distributions from this model. In Figure 3, for each test sequence we display a sample frame and the associated top ten matches from the training data according to the score (4). We can see how the pose is matched against change of appearance and motion style, specifically a test subject pose is matched to similar poses of different subjects in the training set. This shows how the topic representation factors out most of the appearance variations and retains only essential information on the pose.

In order to give a quantitative evaluation of the pose matching performance and compare with other approaches, we labeled the dataset by mapping the set of walking poses to the interval $[0, 1]$. We manually assigned 0 to the frames at the beginning of the double support phase, when the swinging

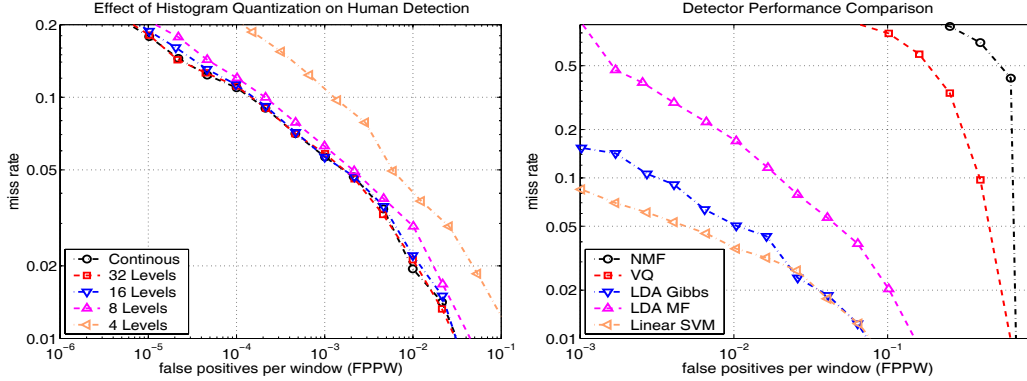

Figure 1: **Human detection results.** (Left) Effect on human detection performances of quantizing the histogram of oriented gradient descriptor [6] for a boosted linear SVM classifier based on these features. Here we show false positive vs. false negative curves on log scale. We can see that for 16 quantization levels or more the differences are negligible, thus validating our discrete approach. (Right) Performances of five detectors using HOG features trained without boosting and tested on the INRIA dataset: LDA detectors learned by Gibbs Sampling and Mean Field, Vector Quantization, Non-negative Matrix Factorization - all with $K = 24$ components/codewords - and Linear SVM. We can see how the Gibbs LDA outperform by far the other unsupervised clustering techniques and scores comparably with the Linear SVM, which is specifically optimized for the simpler binary classification problem.

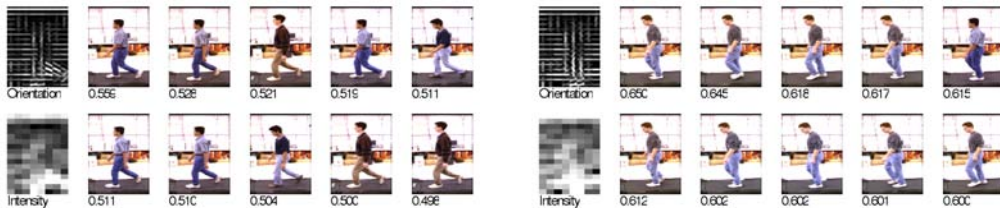

Figure 2: **Topics distributions and clusters.** We show sample topics (2 out of 8) from the LDA model trained on the single view Mobo sequences. For each topic $k$, we show 12 images in 2 rows. The first column shows the distribution of local orientations associated with topic $k$: (top) visualization of the orientations and (bottom) average gradient intensities for each cell. The right 5 columns show the top ten images in the dataset with highest topic proportion $\hat{\theta}_k$, shown below each image. We can see that topics are tightly related to pose classes.

foot touches the ground, and 1 to the frames where the legs are approximately parallel. We labeled the remaining frames automatically using linear interpolation between keyframes. The average interval between keyframes is 8.1 frames, this motivates our choice of the number of topics $K = 8$. For each test frame, we computed the pose error as the difference between the associated pose value and the average pose of the best top 10 matches in the training dataset. We obtained an average error of 0.16, corresponding to 1.3 frames. In Figure 4 we show the average pose error per test sequence obtained with our approach compared with Vector Quantization, where the pose is obtained as average of labels associated with the closest clusters, and Non-negative Matrix Factorization, where as in LDA similarity of poses is computed as dot product of the component weights. In all the models we set equal number of components/clusters to $K = 8$. We can see that our approach performs best in all testing sequences. In Figure 4 we also show the average pose error when matching test frames to a single train sequence. Although the different appearance affects the matching performance, overall the results shows how our approach can be successfully applied to automatically match poses of different subjects.

## 6  Conclusions

We introduce a novel approach to human detection, pose classification and matching from a single image. Starting from a representation robust to a limited range of variations in the appearance of humans in images, we derive a generative probabilistic model which allows for automatic discovery of pose information. The model can successfully perform detection and provides a low dimensional representation of the pose. It automatically clusters the images using representative distributions and allows for an efficient approach to pose matching. Our experiments show that our approach matches or exceeds the state of the art in human detection, pose classification and matching.

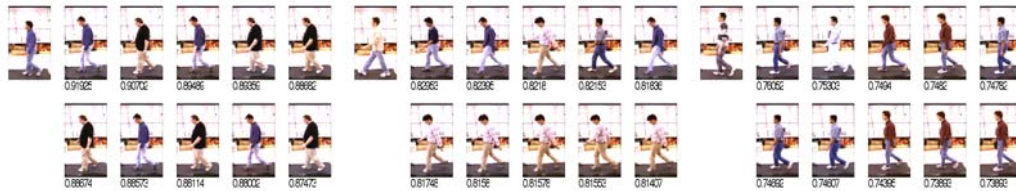

Figure 3: **Pose matching examples.** On the left one sample frame from test sequences, on the right the top 10 matches in the training set based on the similarity score (4), reported below the image. We can see how our approach allows to match poses even despite large changes in appearance, and the same pose is correctly matched across different subjects.

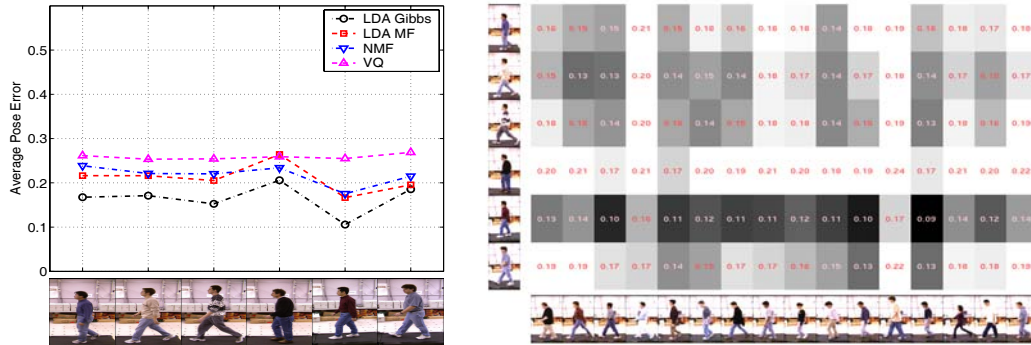

Figure 4: **Pose matching error.** (Left) Average pose error in matching test sequences to the training set, for our model (both Gibbs and Mean Field learning), Non-Negative Matrix Factorization and Vector Quantization. We see how our model trained with Gibbs sampling model clearly outperforms the other approaches. (Right) Average pose error in matching test and training sequence pairs with our approach, where each row is a test sequence and each column a training sequence. The highest error corresponds to about 2 frames, while the mean error is 0.16 and amounts to approximately 1.3 frames.

## Acknowledgments

This work was conducted while the first author was an intern at Honda Research Institute in 2005. Work at UCLA was supported by AFOSR F49620-03-1-0095 and ONR N00014-03-1-0850:P0001.

## Footnotes

[1]Notice that, due to the particular normalization procedure applied, the histogram features we consider here do not have unit norm (in fact, they are zero on uniform regions).

[2]Ideally we would like to use the posterior ratio $R = P(\text{Human}|I_{new})/P(\text{Background}|I_{new})$. However notice that R is equal to (2) if we assume equal priors $P(Human) = P(Background)$.

## References

[1] A. Agarwal and B. Triggs. 3d human pose from silhouettes by relevance vector regression. *CVPR*, 2004.

[2] A. Agarwal and B. Triggs. Hyperfeatures: Multilevel local coding for visual recognition. *ECCV*, 2006.

[3] D. Blei, A. Ng, and M. Jordan. Latent drichlet allocation. *Journal on Machine Learning Research*, 2003.

[4] W. Buntine and A. Jakulin. Discrete principal component analysis. *HIIT Technical Report*, 2005.

[5] J. Canny. GaP: a factor model for discrete data. *ACM SIGIR*, pages 122–129, 2004.

[6] N. Dalal and B. Triggs. Histograms of oriented gradients for human detection. *CVPR*, 2005.

[7] P. F. Felzenszwalb and D. P. Huttenlocher. Efficient matching of pictorial structures. *CVPR*, 2000.

[8] R. Fergus, L. Fei-Fei, P. Perona, and A. Zisserman. Learning object categories from Google's image search. *Proc. ICCV*, pages 1816–1823, 2005.

[9] D. M. Gavrila and V. Philomin. Real-time object detection for smart vehicles. *Proc. ICCV*, 1999.

[10] T. L. Griffiths and M. Steyvers. Finding scientific topics. *Proc. National Academy of Science*, 2004.

[11] R. Gross and J. Shi. The cmu motion of body dataset. Technical report, CMU, 2001.

[12] G.Shakhnarovich, P.Viola, and T.Darrell Fast pose estimation with parameter-sensitive hashing *ICCV*, 2003.

[13] D. Lee and H. Seung. Learning the parts of objects by non-negative matrix factorization. *Nature*, 1999.

[14] D. G. Lowe. Object recognition from local scale-invariant features. *Proc. ICCV*, pages 1150–1157, 1999.

[15] G. Mori, X. Ren, A. A. Efros, and J. Malik. Recovering human body configurations: Combining segmentation and recognition. *Proc. CVPR*, 2:326–333, 2004.

[16] J. C. Niebles, H. Wang, and L. Fei-Fei. Unsupervised learning of human action categories using spatial-temporal words. *Proc. BMVC*, 2006.

[17] K. Nigam, A. K. McCallum, S. Thurn, and T. Mitchell. Text classification from labeled and unlabeled documents using EM. *Machine Learning*, pages 1–34, 2000.

[18] P.Viola, M.Jones, and D.Snow. Detecting pedestrians using patterns of motion and appearance *ICCV*, 2003

[19] R. Ronfard, C. Schmid, and B. Triggs. Learning to parse pictures of people. *ECCV*, 2002.

[20] R. Rosales and S. Sclaroff. Inferring body without tracking body parts. *Proc. CVPR*, 2:506–511, 2000.

[21] L. Sigal, M. Isard, B. H. Sigelman, and M. Black. Attractive people: Assembling loose-limbed models using non-parametric belief propagation. *Proc. NIPS*, pages 1539–1546, 2003.

[22] J. Sivic, B. C. Russell, A. A. Efros, A. Zisserman, and W. T. Freeman. Discovering object categories in image collections. *Proc. ICCV*, 2005.

[23] M. Weber, M. Welling, and P. Perona. Toward automatic discovery of object categories. *CVPR*, 2000.
